# Topographic Transformation as a Discrete Latent Variable

Nebojsa Jojic
Beckman Institute
University of Illinois at Urbana
www.ifp.uiuc.edu/~jojic

Brendan J. Frey
Computer Science
University of Waterloo
www.cs.uwaterloo.ca/~frey

## Abstract

Invariance to topographic transformations such as translation and shearing in an image has been successfully incorporated into feedforward mechanisms, *e.g.*, "convolutional neural networks", "tangent propagation". We describe a way to add transformation invariance to a generative density model by approximating the nonlinear transformation manifold by a discrete set of transformations. An EM algorithm for the original model can be extended to the new model by computing expectations over the set of transformations. We show how to add a discrete transformation variable to Gaussian mixture modeling, factor analysis and mixtures of factor analysis. We give results on filtering microscopy images, face and facial pose clustering, and handwritten digit modeling and recognition.

## 1 Introduction

Imagine what happens to the point in the $N$-dimensional space corresponding to an $N$-pixel image of an object, while the object is deformed by shearing. A very small amount of shearing will move the point only slightly, so deforming the object by shearing will trace a continuous curve in the space of pixel intensities. As illustrated in Fig. 1a, extensive levels of shearing will produce a highly nonlinear curve (consider shearing a thin vertical line), although the curve can be approximated by a straight line locally.

Linear approximations of the transformation manifold have been used to significantly improve the performance of feedforward discriminative classifiers such as nearest neighbors (Simard *et al.*, 1993) and multilayer perceptrons (Simard *et al.*, 1992). Linear generative models (factor analysis, mixtures of factor analysis) have also been modified using linear approximations of the transformation manifold to build in some degree of transformation invariance (Hinton *et al.*, 1997).

In general, the linear approximation is accurate for transformations that couple neighboring pixels, but is inaccurate for transformations that couple nonneighboring pixels. In some applications (*e.g.*, handwritten digit recognition), the input can be blurred so that the linear approximation becomes more robust.

For significant levels of transformation, the nonlinear manifold can be better modeled using a discrete approximation. For example, the curve in Fig. 1a can be

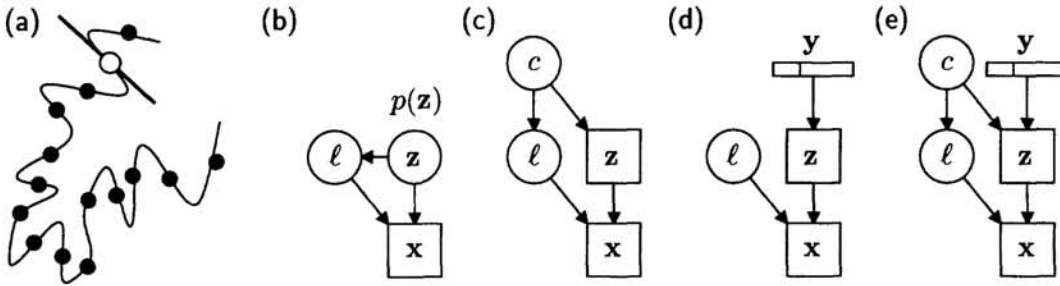

Figure 1: (a) An $N$-pixel greyscale image is represented by a point (unfilled disc) in an $N$-dimensional space. When the object being imaged is deformed by shearing, the point moves along a continuous curve. Locally, the curve is linear, but high levels of shearing produce a highly nonlinear curve, which we approximate by discrete points (filled discs) indexed by $\ell$. (b) A graphical model showing how a discrete transformation variable $\ell$ can be added to a density model $p(\mathbf{z})$ for a *latent image* $\mathbf{z}$ to model the observed image $\mathbf{x}$. The Gaussian pdf $p(\mathbf{x}|\ell, \mathbf{z})$ captures the $\ell$th transformation plus a small amount of pixel noise. (We use a box to represent variables that have Gaussian conditional pdfs.) We have explored (c) transformed mixtures of Gaussians, where $c$ is a discrete cluster index; (d) transformed component analysis (TCA), where $\mathbf{y}$ is a vector of Gaussian factors, some of which may model locally linear transformation perturbations; and (e) mixtures of transformed component analyzers, or transformed mixtures of factor analyzers.

represented by a set of points (filled discs). In this approach, a discrete set of possible transformations is specified beforehand and parameters are learned so that the model is invariant to the set of transformations. This approach has been used to design "convolutional neural networks" that are invariant to translation (Le Cun *et al.*, 1998) and to develop a general purpose learning algorithm for generative topographic maps (Bishop *et al.*, 1998).

We describe how invariance to a discrete set of *known* transformations (like translation) can be built into a generative density model and we show how an EM algorithm for the original density model can be extended to the new model by computing expectations over the set of transformations. We give results for 5 different types of experiment involving translation and shearing.

## 2    Transformation as a Discrete Latent Variable

We represent transformation $\ell$ by a sparse transformation generating matrix $\mathbf{G}_\ell$ that operates on a vector of pixel intensities. For example, integer-pixel translations of an image can be represented by permutation matrices. Although other types of transformation matrix may not be accurately represented by permutation matrices, many useful types of transformation can be represented by sparse transformation matrices. For example, rotation and blurring can be represented by matrices that have a small number of nonzero elements per row (*e.g.*, at most 6 for rotations).

The observed image $\mathbf{x}$ is linked to the nontransformed *latent image* $\mathbf{z}$ and the transformation index $\ell \in \{1, \ldots, L\}$ as follows:

$$p(\mathbf{x}|\ell, \mathbf{z}) = \mathcal{N}(\mathbf{x}; \mathbf{G}_\ell \mathbf{z}, \mathbf{\Psi}), \tag{1}$$

where $\mathbf{\Psi}$ is a diagonal matrix of pixel noise variances. Since the probability of a transformation may depend on the latent image, the joint distribution over the latent image $\mathbf{z}$, the transformation index $\ell$ and the observed image $\mathbf{x}$ is

$$p(\mathbf{x}, \ell, \mathbf{z}) = \mathcal{N}(\mathbf{x}; \mathbf{G}_\ell \mathbf{z}, \mathbf{\Psi}) P(\ell|\mathbf{z}) p(\mathbf{z}). \tag{2}$$

The corresponding graphical model is shown in Fig. 1b. For example, to model noisy transformed images of just one shape, we choose $p(\mathbf{z})$ to be a Gaussian distribution.

**2.1 Transformed mixtures of Gaussians (TMG).** Fig. 1c shows the graphical model for a TMG, where different clusters may have different transformation probabilities. Cluster $c$ has mixing proportion $\pi_c$, mean $\boldsymbol{\mu}_c$ and diagonal covariance matrix $\boldsymbol{\Phi}_c$. The joint distribution is

$$p(\mathbf{x}, \ell, \mathbf{z}, c) = \mathcal{N}(\mathbf{x}; \mathbf{G}_\ell \mathbf{z}, \boldsymbol{\Psi}) \mathcal{N}(\mathbf{z}; \boldsymbol{\mu}_c, \boldsymbol{\Phi}_c) \rho_{\ell c} \pi_c, \tag{3}$$

where the probability of transformation $\ell$ for cluster $c$ is $\rho_{\ell c}$. Marginalizing over the latent image gives the cluster/transformation conditional likelihood,

$$p(\mathbf{x}|\ell, c) = \mathcal{N}(\mathbf{x}; \mathbf{G}_\ell \boldsymbol{\mu}_c, \mathbf{G}_\ell \boldsymbol{\Phi}_c \mathbf{G}_\ell^T + \boldsymbol{\Psi}), \tag{4}$$

which can be used to compute $p(\mathbf{x})$ and the cluster/transformation responsibility $P(\ell, c|\mathbf{x})$. This likelihood looks like the likelihood for a mixture of factor analyzers (Ghahramani and Hinton, 1997). However, whereas the likelihood computation for $N$ latent pixels takes order $N^3$ time in a mixture of factor analyzers, it takes *linear* time, order $N$, in a TMG, because $\mathbf{G}_\ell \boldsymbol{\Phi}_c \mathbf{G}_\ell^T + \boldsymbol{\Psi}$ is sparse.

**2.2 Transformed component analysis (TCA).** Fig. 1d shows the graphical model for TCA (or "transformed factor analysis"). The latent image is modeled using linearly combined Gaussian factors, $\mathbf{y}$. The joint distribution is

$$p(\mathbf{x}, \ell, \mathbf{z}, \mathbf{y}) = \mathcal{N}(\mathbf{x}; \mathbf{G}_\ell \mathbf{z}, \boldsymbol{\Psi}) \mathcal{N}(\mathbf{z}; \boldsymbol{\mu} + \boldsymbol{\Lambda} \mathbf{y}, \boldsymbol{\Phi}) \mathcal{N}(\mathbf{y}; 0, \mathbf{I}) \rho_\ell, \tag{5}$$

where $\boldsymbol{\mu}$ is the mean of the latent image, $\boldsymbol{\Lambda}$ is a matrix of latent image components (the factor loading matrix) and $\boldsymbol{\Phi}$ is a diagonal noise covariance matrix for the latent image. Marginalizing over the factors and the latent image gives the transformation conditional likelihood,

$$p(\mathbf{x}|\ell) = \mathcal{N}(\mathbf{x}; \mathbf{G}_\ell \boldsymbol{\mu}, \mathbf{G}_\ell (\boldsymbol{\Lambda} \boldsymbol{\Lambda}^T + \boldsymbol{\Phi}) \mathbf{G}_\ell^T + \boldsymbol{\Psi}), \tag{6}$$

which can be used to compute $p(\mathbf{x})$ and the transformation responsibility $p(\ell|\mathbf{x})$. $\mathbf{G}_\ell (\boldsymbol{\Lambda} \boldsymbol{\Lambda}^T + \boldsymbol{\Phi}) \mathbf{G}_\ell^T$ is not sparse, so computing this likelihood exactly takes $N^3$ time. However, the likelihood *can* be computed in linear time if we assume $|\mathbf{G}_\ell (\boldsymbol{\Lambda} \boldsymbol{\Lambda}^T + \boldsymbol{\Phi}) \mathbf{G}_\ell^T + \boldsymbol{\Psi}| \approx |\mathbf{G}_\ell (\boldsymbol{\Lambda} \boldsymbol{\Lambda}^T + \boldsymbol{\Phi}) \mathbf{G}_\ell^T|$, which corresponds to assuming that the observed noise is smaller than the variation due to the latent image, or that the observed noise is accounted for by the latent noise model, $\boldsymbol{\Phi}$. In our experiments, this approximation did not lead to degenerate behavior and produced useful models.

By setting columns of $\boldsymbol{\Lambda}$ equal to the derivatives of $\boldsymbol{\mu}$ with respect to continuous transformation parameters, a TCA can accommodate *both* a local linear approximation and a discrete approximation to the transformation manifold.

**2.3 Mixtures of transformed component analyzers (MTCA).** A combination of a TMG and a TCA can be used to jointly model clusters, linear components and transformations. Alternatively, a mixture of Gaussians that is invariant to a discrete set of transformations *and* locally linear transformations can be obtained by combining a TMG with a TCA whose components are all set equal to transformation derivatives.

The joint distribution for the combined model in Fig. 1e is

$$p(\mathbf{x}, \ell, \mathbf{z}, c, \mathbf{y}) = \mathcal{N}(\mathbf{x}; \mathbf{G}_\ell \mathbf{z}, \boldsymbol{\Psi}) \mathcal{N}(\mathbf{z}; \boldsymbol{\mu}_c + \boldsymbol{\Lambda}_c \mathbf{y}, \boldsymbol{\Phi}_c) \mathcal{N}(\mathbf{y}; 0, \mathbf{I}) \rho_{\ell c} \pi_c. \tag{7}$$

The cluster/transformation likelihood is $p(\mathbf{x}|\ell, c) = \mathcal{N}(\mathbf{x}; \mathbf{G}_\ell \boldsymbol{\mu}_c, \mathbf{G}_\ell (\boldsymbol{\Lambda}_c \boldsymbol{\Lambda}_c^T + \boldsymbol{\Phi}_c) \mathbf{G}_\ell^T + \boldsymbol{\Psi})$, which can be approximated in linear time as for TCA.

## 3   Mixed Transformed Component Analysis (MTCA)

We present an EM algorithm for MTCA; EM algorithms for TMG or TCA emerge by setting the number of factors to 0 or setting the number of clusters to 1.

Let $\theta$ represent a parameter in the generative model. For i.i.d. data, the derivative of the log-likelihood of a training set $\mathbf{x}_1, \ldots, \mathbf{x}_T$ with respect to $\theta$ can be written

$$\frac{\partial \log p(\mathbf{x}_1, \ldots, \mathbf{x}_T)}{\partial \theta} = \sum_{t=1}^{T} \mathrm{E}\left[\frac{\partial}{\partial \theta} \log p(\mathbf{x}_t, c, \ell, \mathbf{z}, \mathbf{y}) \big| \mathbf{x}_t\right], \tag{8}$$

where the expectation is taken over $p(c, \ell, \mathbf{z}, \mathbf{y}|\mathbf{x}_t)$. The EM algorithm iteratively solves for a new set of parameters using the old parameters to compute the expectations. This procedure consistently increases the likelihood of the training data.

By setting (8) to 0 and solving for the new parameter values, we obtain update equations based on the expectations given in the Appendix. Notation: $\langle \cdot \rangle = \frac{1}{T}\sum_{t=1}^{T}(\cdot)$ is a sufficient statistic computed by averaging over the training set; $\mathrm{diag}(\mathbf{A})$ gives a vector containing the diagonal elements of matrix $\mathbf{A}$; $\mathrm{diag}(\mathbf{a})$ gives a diagonal matrix whose diagonal contains the elements of vector $\mathbf{a}$; and $\mathbf{a} \circ \mathbf{b}$ gives the element-wise product of vectors $\mathbf{a}$ and $\mathbf{b}$. Denoting the updated parameters by "~", we have

$$\tilde{\pi}_c = \langle P(c|\mathbf{x}_t)\rangle, \qquad \tilde{\rho}_{\ell c} = \langle P(\ell|\mathbf{x}_t, c)\rangle, \tag{9}$$

$$\tilde{\boldsymbol{\mu}}_c = \frac{\langle P(c|\mathbf{x}_t)\mathrm{E}[\mathbf{z} - \boldsymbol{\Lambda}_c\mathbf{y}|\mathbf{x}_t, c]\rangle}{\langle P(c|\mathbf{x}_t)\rangle}, \tag{10}$$

$$\tilde{\boldsymbol{\Phi}}_c = \frac{\mathrm{diag}\left(\langle P(c|\mathbf{x}_t)\mathrm{E}[(\mathbf{z}-\boldsymbol{\mu}_c-\boldsymbol{\Lambda}_c\mathbf{y})\circ(\mathbf{z}-\boldsymbol{\mu}_c-\boldsymbol{\Lambda}_c\mathbf{y})|\mathbf{x}_t, c]\rangle\right)}{\langle P(c|\mathbf{x}_t)\rangle}, \tag{11}$$

$$\tilde{\boldsymbol{\Psi}} = \mathrm{diag}\left(\langle \mathrm{E}[(\mathbf{x}_t-\mathbf{G}_\ell\mathbf{z})\circ(\mathbf{x}_t-\mathbf{G}_\ell\mathbf{z})|\mathbf{x}_t]\rangle\right), \tag{12}$$

$$\tilde{\boldsymbol{\Lambda}}_c = \langle P(c|\mathbf{x}_t)\mathrm{E}[(\mathbf{z} - \boldsymbol{\mu}_c)\mathbf{y}^{\mathrm{T}}|\mathbf{x}_t]\rangle\langle P(c|\mathbf{x}_t)\mathrm{E}[\mathbf{y}\mathbf{y}^{\mathrm{T}}|\mathbf{x}_t]\rangle^{-1}. \tag{13}$$

To reduce the number of parameters, we will sometimes assume $\rho_{\ell c}$ does not depend on $c$ or even that $\rho_{\ell c}$ is held constant at a uniform distribution.

## 4   Experiments

### 4.1   Filtering Images from a Scanning Electron Microscope (SEM).
SEM images (e.g., Fig. 2a) can have a very low signal to noise ratio due to a high variance in electron emission rate and modulation of this variance by the imaged material (Golem and Cohen, 1998). To reduce noise, multiple images are usually averaged and the pixel variances can be used to estimate certainty in rendered structures. Fig. 2b shows the estimated means and variances of the pixels from 230 140 × 56 SEM images like the ones in Fig. 2a. In fact, averaging images does not take into account spatial uncertainties and filtering in the imaging process introduced by the electron detectors and the high-speed electrical circuits.

We trained a single-cluster TMG with 5 horizontal shifts and 5 vertical shifts on the 230 SEM images using 30 iterations of EM. To keep the number of parameters almost equal to the number of parameters estimated using simple averaging, the transformation probabilities were not learned and the pixel variances in the observed image were set equal after each M step. So, TMG had 1 more parameter. Fig. 2c shows the mean and variance learned by the TMG. Compared to simple averaging, the TMG finds sharper, more detailed structure. The variances are significantly lower, indicating that the TMG produces a more confident estimate of the image.

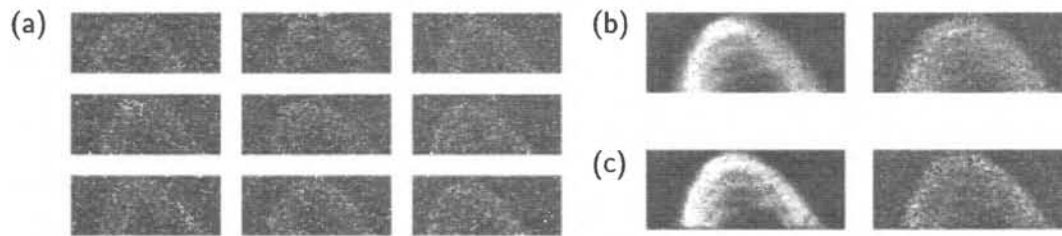

Figure 2: (a) 140 × 56 pixel SEM images. (b) The mean and variance of the image pixels. (c) The mean and variance found by a TMG reveal more structure and less uncertainty.

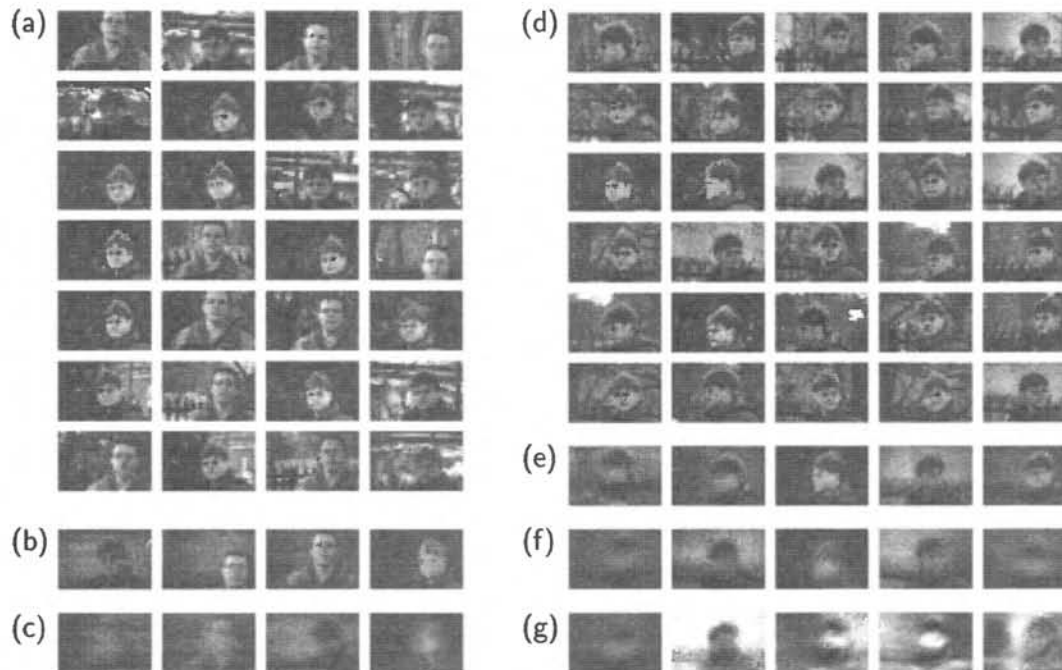

Figure 3: (a) Frontal face images of two people. (b) Cluster means learned by a TMG and (c) a mixture of Gaussians. (d) Images of one person with different poses. (e) Cluster means learned by a TMG. (f) Less detailed cluster means learned by a mixture of Gaussians. (g) Mean and first 4 principal components of the data, which mostly model lighting and translation.

**4.2 Clustering Faces and Poses.** Fig. 3a shows examples from a training set of 400 jerky images of two people walking across a cluttered background. We trained a TMG with 4 clusters, 11 horizontal shifts and 11 vertical shifts using 15 iterations of EM after initializing the weights to small, random values. The loop-rich MATLAB script executed in 40 minutes on a 500MHz Pentium processor. Fig. 3b shows the cluster means, which include two sharp representations of each person's face, with the background clutter suppressed. Fig. 3c shows the much blurrier means for a mixture of Gaussians trained using 15 iterations of EM.

Fig. 3d shows examples from a training set of 400 jerky images of one person with different poses. We trained a TMG with 5 clusters, 11 horizontal shifts and 11 vertical shifts using 40 iterations of EM. Fig. 3e shows the cluster means, which capture 4 poses and mostly suppress the background clutter. The mean for cluster 4 includes part of the background, but this cluster also has a low mixing proportion of 0.1. A traditional mixture of Gaussians trained using 40 iterations of EM finds blurrier means, as shown in Fig. 3f. The first 4 principal components mostly try to account for lighting and translation, as shown in Fig. 3g.

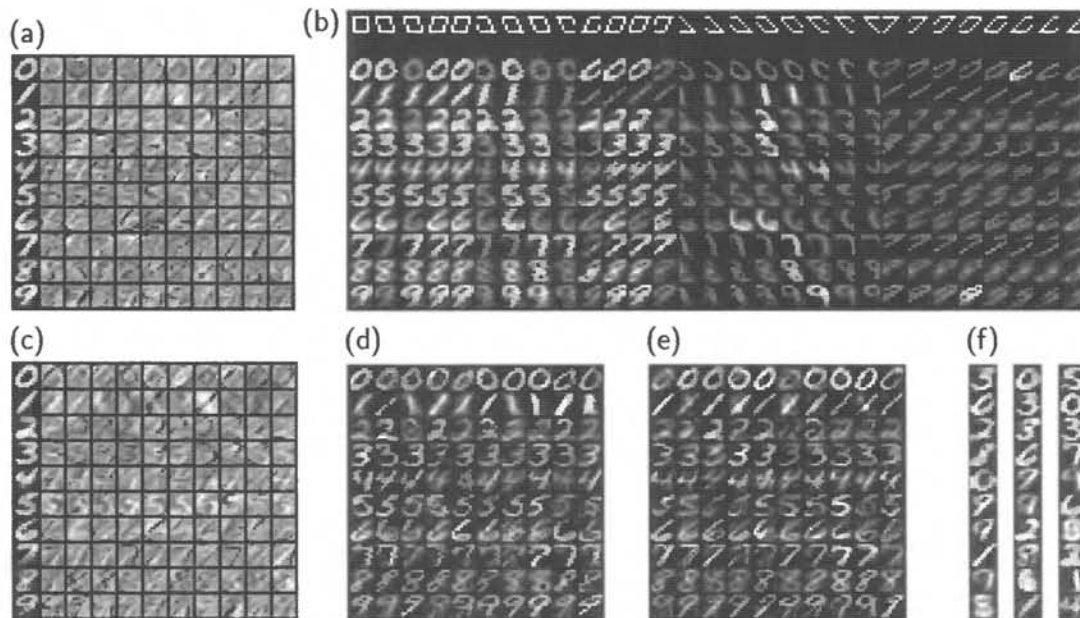

Figure 4: Modeling handwritten digits. (a) Means and components and (b) the sheared + translated means (dimmed transformations have low probability) for each of 10 TCA models trained on 200 examples of each digit. (c) Means and components of 10 FA models trained on the same data. (d) Digits generated from the 10 TCA models and (e) the 10 FA models. (f) The means for a mixture of 10 Gaussians, a mixture of 10 factor analyzers and a 10-cluster TMG trained on all 2000 digits. In each case, the best of 10 experiments was selected.

**4.3 Modeling Handwritten Digits.** We performed both supervised and unsupervised learning experiments on $8 \times 8$ greyscale versions of 2000 digits from the CEDAR CDROM (Hull, 1994). Although the preprocessed images fit snugly in the $8 \times 8$ window, there is wide variation in "writing angle" (*e.g.*, the vertical stroke of the 7 is at different angles). So, we produced a set of 29 shearing+translation transformations (see the top row of Fig. 4b) to use in transformed density models.

In our supervised learning experiments, we trained one 10-component TCA on each class of digit using 30 iterations of EM. Fig. 4a shows the mean and 10 components for each of the 10 models. The lower 10 rows of images in Fig. 4b show the sheared and translated means. In cases where the transformation probability is below 1%, the image is dimmed. We also trained one 10-component factor analyzer on each class of digit using 30 iterations of EM. The means and components are shown in Fig. 4c. The means found by TCA are sharper and whereas the components found by factor analysis often account for writing angle (*e.g.*, see the components for 7) the components found by TCA tend to account for line thickness and arc size. Fig. 4d and e show digits that were randomly generated from the TCAs and the factor analyzers. Since different components in the factor analyzers account for different stroke angles, the simulated digits often have an extra stroke, whereas digits simulated from the TCAs contain fewer spurious strokes.

To test recognition performance, we trained 10-component factor analyzers and TCAs on 200 examples of each digit using 50 iterations of EM. Each set of models used Bayes rule to classify 1000 test patterns and while factor analysis gave an error rate of 3.2%, TCA gave an error rate of only 2.7%.

In our unsupervised learning experiments, we fit 10-cluster mixture models to the entire set of 2000 digits to see which models could identify all 10 digits. We tried a mixture of 10 Gaussians, a mixture of 10 factor analyzers and a 10-cluster TMG. In each case, 10 models were trained using 100 iterations of EM and the model with

the highest likelihood was selected and is shown in Fig. 4f. Compared to the TMG, the first two methods found blurred and repeated classes. After identifying each cluster with its most prevalent class of digit, we found that the first two methods had error rates of 53% and 49%, but the TMG had a much lower error rate of 26%.

## 5 Summary

In many learning applications, we know beforehand that the data includes transformations of an easily specified nature (*e.g.*, shearing of digit images). If a generative density model is learned from the data, the model must extract a model of both the transformations and the more interesting and potentially useful structure. We described a way to add transformation invariance to a generative density model by approximating the transformation manifold with a discrete set of points. This releases the generative model from needing to model the transformations. 5 different types of experiment show that the method is effective and quite efficient.

Although the time needed by this method scales exponentially with the dimensionality of the transformation manifold, we believe that it will be useful in many practical applications and that it illustrates what is possible with a generative model that incorporates a latent transformation variable. We are exploring the performance of a faster variational learning method and extending the model to time series.

**Acknowledgements.** We used CITO, NSERC, NSF and Beckman Foundation grants.

**References**

C. M. Bishop, M. Svensen and C. K. I. Williams 1998. GTM: The generative topographic mapping. *Neural Computation* 10:1, 215–235.

G. E. Hinton, P. Dayan and M. Revow 1997. Modeling the manifolds of images of handwritten digits. *IEEE Trans. on Neural Networks* 8, 65–74.

Z. Ghahramani and G. E. Hinton 1997. The EM algorithm for mixtures of factor analyzers. University of Toronto Technical Report CRG-TR-96-1. Available at www.gatsby.ucl.ac.uk/~zoubin.

R. Golem and I. Cohen 1998. Scanning electron microscope image enhancement. School of Computer and Electrical Engineering project report, Ben-Gurion University.

J. J. Hull 1994. A database for handwritten text recognition research. *IEEE Trans. on Pattern Analysis and Machine Intelligence* 16:5, 550–554.

Y. Le Cun, L. Bottou, Y. Bengio and P. Haffner 1998. Gradient-based learning applied to document recognition. *Proceedings of the IEEE* 86:11, November, 2278–2324.

P. Y. Simard, B. Victorri, Y. Le Cun and J. Denker 1992. Tangent Prop – A formalism for specifying selected invariances in an adaptive network. In *Advances in Neural Information Processing Systems 4*, Morgan Kaufmann, San Mateo, CA.

P. Y. Simard, Y. Le Cun and J. Denker 1993. Efficient pattern recognition using a new transformation distance. In S. J. Hanson, J. D. Cowan and C. L. Giles, *Advances in Neural Information Processing Systems 5*, Morgan Kaufmann, San Mateo, CA.

**Appendix: The Sufficient Statistics Found in the E-Step**

The sufficient statistics for the M-Step are computed in the E-Step using sparse linear algebra during a single pass through the training set. Before making this pass, the following matrices are computed: $\Omega_{\ell,c} = \text{COV}(\mathbf{z}|\mathbf{x},\mathbf{y},\ell,c) = (\Phi_c^{-1} + \mathbf{G}_\ell'\Psi^{-1}\mathbf{G}_\ell)^{-1}$, $\beta_{\ell,c} = \text{COV}(\mathbf{y}|\mathbf{x},\ell,c) = (\mathbf{I} + \Lambda_c'\Phi_c^{-1}\Lambda_c - \Lambda_c'\Phi_c^{-1}\Omega_{\ell,c}\Phi_c^{-1}\Lambda_c)^{-1}$. For each case in the training set, $P(c,\ell|\mathbf{x}_t)$ is first computed for each combination of $c,\ell$, before computing $\text{E}[\mathbf{y}|\mathbf{x}_t,\ell,c] = \beta_{\ell,c}\Lambda_c'\Phi_c^{-1}[\Omega_{\ell,c}\mathbf{G}_\ell'\Psi^{-1}\mathbf{x}_t - (\mathbf{I}-\Omega_{\ell,c}\Phi_c^{-1})\mu_c]$, $\text{E}[\mathbf{z}|\mathbf{x}_t,\ell,c] = \mu_c + \Omega_{\ell,c}\mathbf{G}_\ell'\Psi^{-1}(\mathbf{x}_t - \mathbf{G}_\ell\mu_c) + \Omega_\ell\Phi^{-1}\Lambda_c\beta_{\ell,c}\Lambda_c'\Phi_c^{-1}\Omega_{\ell,c}\mathbf{G}_\ell'\Psi^{-1}(\mathbf{x}_t - \mathbf{G}_\ell\mu_c)$, $\text{E}[(\mathbf{z}-\mu_c)(\mathbf{z}-\mu_c)|\mathbf{x}_t,\ell,c] = (\text{E}[\mathbf{z}|\mathbf{x}_t,\ell,c]-\mu_c)(\text{E}[\mathbf{z}|\mathbf{x}_t,\ell,c]-\mu_c) + \text{diag}(\Omega_{\ell,c}) + \text{diag}(\Omega_{\ell,c}\Phi_c^{-1}\Lambda_c\beta_{\ell,c}\Lambda_c'\Phi_c^{-1}\Omega_{\ell,c})$, $\text{E}[(\mathbf{z}-\mu_c)\mathbf{y}'|\mathbf{x}_t,\ell,c] = (\text{E}[\mathbf{z}|\mathbf{x}_t,\ell,c] - \mu_c)\text{E}[\mathbf{y}|\mathbf{x}_t,\ell,c]' + \Omega_{\ell,c}\Phi_c^{-1}\Lambda_c\beta_{\ell,c}$. The expectations needed in (10)-(13) are then computed from $P(c|\mathbf{x}_t)\text{E}[\mathbf{z} - \Lambda_c\mathbf{y}|\mathbf{x}_t,c] = \sum_\ell P(c,\ell|\mathbf{x}_t)(\text{E}[\mathbf{z}|\mathbf{x}_t,\ell,c] - \Lambda_c\text{E}[\mathbf{y}|\mathbf{x}_t,\ell,c])$, $P(c|\mathbf{x}_t)\text{E}[(\mathbf{z}-\mu_c - \Lambda_c\mathbf{y})\circ(\mathbf{z}-\mu_c-\Lambda_c\mathbf{y})|\mathbf{x}_t,c] = \sum_\ell P(c,\ell|\mathbf{x}_t)\{\text{E}[(\mathbf{z}-\mu_c)\circ(\mathbf{z}-\mu_c)|\mathbf{x}_t,\ell,c] + \text{diag}(\Lambda_c\beta_{\ell,c}\Lambda_c') - 2\text{diag}(\Lambda_c\text{E}[(\mathbf{z}-\mu_c)\mathbf{y}'|\mathbf{x}_t,\ell,c]') + (\Lambda_c\text{E}[\mathbf{y}|\mathbf{x}_t,\ell,c])\circ(\Lambda_c\text{E}[\mathbf{y}|\mathbf{x}_t,\ell,c])\}$, $\text{E}[(\mathbf{x}_t-\mathbf{G}_\ell\mathbf{z})\circ(\mathbf{x}_t-\mathbf{G}_\ell\mathbf{z})|\mathbf{x}_t] = \sum_{c,\ell} P(c,\ell|\mathbf{x}_t)\{(\mathbf{x}_t - \mathbf{G}_\ell\text{E}[\mathbf{z}|\mathbf{x}_t,\ell,c]) \circ (\mathbf{x}_t - \mathbf{G}_\ell\text{E}[\mathbf{z}|\mathbf{x}_t,\ell,c]) + \text{diag}(\mathbf{G}_\ell\Omega_{\ell,c}\mathbf{G}_\ell') + \text{diag}(\mathbf{G}_\ell\Omega_{\ell,c}\Phi_c^{-1}\Lambda_c\beta_{\ell,c}\Lambda_c'\Phi_c^{-1}\Omega_{\ell,c}\mathbf{G}_\ell')\}$, $P(c|\mathbf{x}_t)\text{E}[(\mathbf{z}-\mu)\mathbf{y}'|\mathbf{x}_t,c] = \sum_\ell P(c,\ell|\mathbf{x}_t)\text{E}[(\mathbf{z}-\mu)\mathbf{y}'|\mathbf{x}_t,\ell,c]$, $P(c|\mathbf{x}_t)\text{E}[\mathbf{y}\mathbf{y}'|\mathbf{x}_t,c] = \sum_\ell P(c,\ell|\mathbf{x}_t)\beta_{\ell,c} + \sum_\ell P(c,\ell|\mathbf{x}_t)\text{E}[\mathbf{y}|\mathbf{x}_t,\ell,c]\text{E}[\mathbf{y}|\mathbf{x}_t,\ell,c]'$.